# An ideal observer model of infant object perception

**Charles Kemp**
Department of Psychology
Carnegie Mellon University
ckemp@cmu.edu

**Fei Xu**
Department of Psychology
University of British Columbia
fei@psych.ubc.ca

## Abstract

Before the age of 4 months, infants make inductive inferences about the motions of physical objects. Developmental psychologists have provided verbal accounts of the knowledge that supports these inferences, but often these accounts focus on categorical rather than probabilistic principles. We propose that infant object perception is guided in part by probabilistic principles like *persistence*: things tend to remain the same, and when they change they do so gradually. To illustrate this idea we develop an ideal observer model that incorporates probabilistic principles of rigidity and inertia. Like previous researchers, we suggest that rigid motions are expected from an early age, but we challenge the previous claim that the inertia principle is relatively slow to develop [1]. We support these arguments by modeling several experiments from the developmental literature.

Over the past few decades, ingenious experiments [1, 2] have suggested that infants rely on systematic expectations about physical objects when interpreting visual scenes. Looking time studies suggest, for example, that infants expect objects to follow continuous trajectories through time and space, and understand that two objects cannot simultaneously occupy the same location. Many of these studies have been replicated several times, but there is still no consensus about the best way to characterize the knowledge that gives rise to these findings.

Two main approaches can be found in the literature. The *verbal* approach uses natural language to characterize principles of object perception [1, 3]: for example, Spelke [4] proposes that object perception is consistent with principles including *continuity* ("a moving object traces exactly one connected path over space and time") and *cohesion* ("a moving object maintains its connectedness and boundaries"). The *mechanistic* approach proposes that physical knowledge is better characterized by describing the mechanisms that give rise to behavior, and researchers working in this tradition often develop computational models that support their theoretical proposals [5]. We pursue a third approach—the *ideal observer* approach [6, 7, 8]—that combines aspects of both previous traditions. Like the verbal approach, our primary goal is to characterize principles that account for infant behavior, and we will not attempt to characterize the mechanisms that produce this behavior. Like the mechanistic approach, we emphasize the importance of formal models, and suggest that these models can capture forms of knowledge that are difficult for verbal accounts to handle.

Ideal observer models [6, 9] specify the conclusions that normatively follow given a certain source of information and a body of background knowledge. These models can therefore address questions about the *information* and the *knowledge* that support perception. Approaches to the *information* question characterize the kinds of perceptual information that human observers use. For example, Geisler [9] discusses which components of the information available at the retina contribute to visual perception, and Banks and Shannon [10] use ideal observer models to study the perceptual consequences of immaturities in the retina. Approaches to the *knowledge* question characterize the background assumptions that are combined with the available input in order to make inductive inferences. For example, Weiss and Adelson [7] describe several empirical phenomena that are consistent with the a priori assumption that motions tend to be slow and smooth. There are few previous attempts to develop ideal observer models of infant perception, and most of them focus only on the

information question [10]. This paper addresses the knowledge question, and proposes that the ideal observer approach can help to identify the minimal set of principles needed to account for the visual competence of young infants.

Most verbal theories of object perception focus on categorical principles [4], or principles that make a single distinction between possible and impossible scenes. We propose that physical knowledge in infancy is also characterized by probabilistic principles, or expectations that make some possible scenes more surprising than others. We demonstrate the importance of probabilistic principles by focusing on two examples: the *rigidity* principle states that objects usually maintain their shape and size when they move, and the *inertia* principle states that objects tend to maintain the same pattern of motion over time. Both principles capture important regularities, but exceptions to these regularities are relatively common.

Focusing on rigidity and inertia allows us to demonstrate two contributions that probabilistic approaches can make. First, probabilistic approaches can reinforce current proposals about infant perception. Spelke [3] suggests that rigidity is a core principle that guides object perception from a very early age, and we demonstrate how this idea can be captured by a model that also tolerates exceptions, such as non-rigid biological motion. Second, probabilistic approaches can identify places where existing proposals may need to be revised. Spelke [3] argues that the principle of inertia is slow to develop, but we suggest that a probabilistic version of this principle can help to account for inferences made early in development.

## 1  An ideal observer approach

An ideal observer approach to object perception can be formulated in terms of a generative model for scenes. Scenes can be generated in three steps. First we choose the number $n$ of objects that will appear in the scene, and generate the shape, visual appearance, and initial location of each object. We then choose a velocity field for each object which specifies how the object moves and changes shape over time. Finally, we create a visual scene by taking a two-dimensional projection of the moving objects generated in the two previous steps. An ideal observer approach explores the idea that the inferences made by infants approximate the optimal inferences with respect to this generative model.

We work within this general framework but make two simplifications. We will not discuss how the shapes and visual appearances of objects are generated, and we make the projection step simple by working with a two-dimensional world. These simplifications allow us to focus on the expectations about velocity fields that guide motion perception in infants. The next two sections present two prior distributions that can be used to generate velocity fields. The first is a baseline prior that does not incorporate probabilistic principles, and the second incorporates probabilistic versions of rigidity and inertia. The two priors capture different kinds of knowledge, and we argue that the second provides the more accurate characterization of the knowledge that infants bring to object perception.

### 1.1  A baseline prior on velocity fields

Our baseline prior is founded on five categorical principles that are closely related to principles discussed by Spelke [3, 4]. The principles we consider rely on three basic notions: space, time, and matter. We also refer to *particles*, which are small pieces of matter that occupy space-time points. Particles satisfy several principles:

> C1. *Temporal continuity*. Particles are not created or destroyed. In other words, every particle that exists at time $t_1$ must also exist at time $t_2$.
> C2. *Spatial continuity*. Each particle traces a continuous trajectory through space.
> C3. *Exclusion*. No two particles may occupy the same space-time point.

An object is a collection of particles, and these collections satisfy two principles:

> C4. *Discreteness*. Each particle belongs to exactly one object.
> C5. *Cohesion*. At each point in time, the particles belonging to an object occupy a single connected region of space.

Suppose that we are interested in a space-time window specified by a bounded region of space and a bounded interval of time. For simplicity, we will assume that space is two-dimensional, and that the space-time window corresponds to the unit cube. Suppose that a velocity field $\vec{v}$ assigns a velocity

$(v_x, v_y)$ to each particle in the space-time window, and let $\vec{v_i}$ be the field created by considering only particles that belong to object $i$. We develop a theory of object perception by defining a prior distribution $p(\vec{v})$ on velocity fields.

Consider first the distribution $p(\vec{v_1})$ on fields for a single object. Any field that violates one or more of principles C1–C5 is assigned zero probability. For instance, fields where part of an object winks out of existence violate the principle of temporal continuity, and fields where an object splits into two distinct pieces violate the principle of cohesion. Many fields, however, remain, including fields that specify non-rigid motions and jagged trajectories. For now, assume that we are working with a space of fields that is bounded but very large, and that the prior distribution over this space is uniform for all fields consistent with principles C1–C5:

$$p(\vec{v_1}) \propto f(\vec{v_1}) = \begin{cases} 0 & \text{if } \vec{v_1} \text{ violates C1–C5} \\ 1 & \text{otherwise.} \end{cases} \quad (1)$$

Consider now the distribution $p(\vec{v_1}, \vec{v_2})$ on fields for pairs of objects. Principles C1 through C5 rule out some of these fields, but again we must specify a prior distribution on those that remain. Our prior is induced by the following principle:

C6. *Independence*. Velocity fields for multiple objects are independently generated subject to principles C1 through C5.

More formally, the independence principle specifies how the prior for the multiple object case is related to the prior $p(\vec{v_1})$ on velocity fields for a single object (Equation 1):

$$p(\vec{v_1}, \ldots, \vec{v_n}) \propto f(\vec{v_1}, \ldots, \vec{v_n}) = \begin{cases} 0 & \text{if } \{\vec{v_i}\} \text{ collectively violate C1–C5} \\ f(\vec{v_1}) \ldots f(\vec{v_n}) & \text{otherwise.} \end{cases} \quad (2)$$

## 1.2 A smoothness prior on velocity fields

We now develop a prior $p(\vec{v})$ that incorporates probabilistic expectations about the motion of physical objects. Consider again the prior $p(\vec{v_1})$ on the velocity field $\vec{v_1}$ of a single object. Principles C1–C5 make a single cut that distinguishes possible from impossible fields, but we need to consider whether infants have additional knowledge that makes some of the possible fields less surprising than others. One informal idea that seems relevant is the notion of *persistence*[11]: things tend to remain the same, and when they change they do so gradually. We focus on two versions of this idea that may guide expectations about velocity fields:

S1. *Spatial smoothness*. Velocity fields tend to be smooth in space.
S2. *Temporal smoothness*. Velocity fields tend to be smooth in time.

A field is "smooth in space" if neighboring particles tend to have similar velocities at any instant of time. The smoothest possible field will be one where all particles have the same velocity at any instant—in other words, where an object moves rigidly. The principle of spatial smoothness therefore captures the idea that objects tend to maintain the same shape and size.

A field is "smooth in time" if any particle tends to have similar velocities at nearby instants of time. The smoothest possible field will be one where each particle maintains the same velocity throughout the entire interval of interest. The principle of temporal smoothness therefore captures the idea that objects tend to maintain their initial pattern of motion. For instance, stationary objects tend to remain stationary, moving objects tend to keep moving, and a moving object following a given trajectory tends to continue along that trajectory.

Principles S1 and S2 are related to two principles— *rigidity* and *inertia*—that have been discussed in the developmental literature. The rigidity principle states that objects "tend to maintain their size and shape over motion"[3], and the inertia principle states that objects move smoothly in the absence of obstacles [4]. Some authors treat these principles rather differently: for instance, Spelke suggests that rigidity is one of the core principles that guides object perception from a very early age [3], but that the principle of inertia is slow to develop and is weak or fragile once acquired. Since principles S1 and S2 seem closely related, the suggestion that one develops much later than the other seems counterintuitive. The rest of this paper explores the idea that both of these principles are needed to characterize infant perception.

Our arguments will be supported by formal analyses, and we therefore need formal versions of S1 and S2. There may be different ways to formalize these principles, but we present a simple

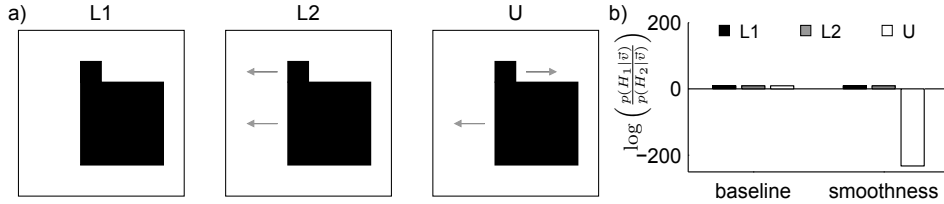

Figure 1: (a) Three scenes inspired by the experiments of Spelke and colleagues [12, 13]. Each scene can be interpreted as a single object, or as a small object on top of a larger object. (b) Relative preferences for the one-object and two-object interpretations according to two models. The baseline model prefers the one-object interpretation in all three cases, but the smoothness model prefers the one-object interpretation only for scenes L1 and L2.

approach that builds on existing models of motion perception in adults [7, 8]. We define measures of instantaneous roughness that capture how rapidly a velocity field $\vec{v}$ varies in space and time:

$$R_{\text{space}}(\vec{v}, t) = \frac{1}{\text{vol}(O(t))} \int_{O(t)} \left| \frac{\partial \vec{v}(x, y, t)}{\partial x} \right|^2 + \left| \frac{\partial \vec{v}(x, y, t)}{\partial y} \right|^2 dx dy \tag{3}$$

$$R_{\text{time}}(\vec{v}, t) = \frac{1}{\text{vol}(O(t))} \int_{O(t)} \left| \frac{\partial \vec{v}(x, y, t)}{\partial t} \right|^2 dx dy \tag{4}$$

where $O(t)$ is the set of all points that are occupied by the object at time $t$, and $\text{vol}(O(t))$ is the volume of the object at time $t$. $R_{\text{space}}(\vec{v}, t)$ will be large if neighboring particles at time $t$ tend to have different velocities, and $R_{\text{time}}(\vec{v}, t)$ will be large if many particles are accelerating at time $t$.

We combine our two roughness measures to create a single smoothness function $S(\cdot)$ that measures the smoothness of a velocity field:

$$S(\vec{v}) = -\lambda_{\text{space}} \int R_{\text{space}}(\vec{v}, t) dt - \lambda_{\text{time}} \int R_{\text{time}}(\vec{v}, t) dt \tag{5}$$

where $\lambda_{\text{space}}$ and $\lambda_{\text{time}}$ are positive weights that capture the importance of spatial smoothness and temporal smoothness. For all analyses in this paper we set $\lambda_{\text{space}} = 10000$ and $\lambda_{\text{time}} = 250$, which implies that violations of spatial smoothness are penalized more harshly than violations of temporal smoothness. We now replace Equation 1 with a prior on velocity fields that takes smoothness into account:

$$p(\vec{v_1}) \propto f(\vec{v_1}) = \begin{cases} 0 & \text{if } \vec{v_1} \text{ violates C1–C5} \\ \exp\left(S(\vec{v_1})\right) & \text{otherwise.} \end{cases} \tag{6}$$

Combining Equation 6 with Equation 2 specifies a model of object perception that incorporates probabilistic principles of rigidity and inertia.

## 2 Empirical findings: spatial smoothness

There are many experiments where infants aged 4 months and younger appear to make inferences that are consistent with the principle of rigidity. This section suggests that the principle of spatial smoothness can account for these results. We therefore propose that a probabilistic principle (spatial smoothness) can explain all of the findings previously presented in support of a categorical principle (rigidity), and can help in addition to explain how infants perceive non-rigid motion.

One set of studies explores inferences about the number of objects in a scene. When a smaller block is resting on top of a larger block (L1 in Figure 1a), 3-month-olds infer that the scene includes a single object [12]. The same result holds when the small and large blocks are both moving in the same direction (L2 in Figure 1a) [13]. When these blocks are moving in opposite directions (U in Figure 1a), however, infants appear to infer that the scene contains two objects [13]. Results like these suggest that infants may have a default expectation that objects tend to move rigidly.

We compared the predictions made by two models about the scenes in Figure 1a. The *smoothness* model uses a prior $p(\vec{v_1})$ that incorporates principles S1 and S2 (Equation 6), and the *baseline* model is identical except that it sets $\lambda_{\text{space}} = \lambda_{\text{time}} = 0$. Both models therefore incorporate principles C1–C6, but only the smoothness model captures the principle of spatial smoothness.

Given any of the scenes in Figure 1a, an infant must solve two problems: she must compute the velocity field $\vec{v}$ for the scene and must decide whether this field specifies the motion of one or two objects. Here we focus on the second problem, and assume that the infant's perceptual system has already computed a veridical velocity field for each scene that we consider. In principle, however, the smoothness prior in Equation 6 can address both problems. Previous authors have shown how smoothness priors can be used to compute velocity fields given raw image data [7, 8].

Let $H_1$ be the hypothesis that a given velocity field corresponds to a single object, and let $H_2$ be the hypothesis that the field specifies the motions of two objects. We assume that the prior probabilities of these hypotheses are equal, and that $P(H_1) = P(H_2) = 0.5$. An ideal observer can use the posterior odds ratio to choose between these hypotheses:

$$\frac{P(H_1|\vec{v})}{P(H_2|\vec{v})} = \frac{P(\vec{v}|H_1)}{P(\vec{v}|H_2)} \frac{P(H_1)}{P(H_2)} \approx \frac{f(\vec{v})}{\int f(\vec{v_1})d\vec{v_1}} \frac{\int f(\vec{v_1},\vec{v_2})d\vec{v_1}d\vec{v_2}}{f(\vec{v_A},\vec{v_B})} \tag{7}$$

Equation 7 follows from Equations 2 and 6, and from approximating $P(\vec{v}|H_2)$ by considering only the two object interpretation $(\vec{v_A}, \vec{v_B})$ with maximum posterior probability. For each scene in Figure 1a, the best two object interpretation will specify a field $\vec{v_A}$ for the small upper block, and a field $\vec{v_B}$ for the large lower block.

To approximate the posterior odds ratio in Equation 7 we compute rough approximations of $\int f(\vec{v_1})d\vec{v_1}$ and $\int f(\vec{v_1},\vec{v_2})d\vec{v_1}d\vec{v_2}$ by summing over a finite space of velocity fields. As described in the supporting material, we consider all fields that can be built from objects with 5 possible shapes, 900 possible starting locations, and 10 possible trajectories. For computational tractability, we convert each continuous velocity field to a discrete field defined over a space-time grid with 45 cells along each spatial dimension and 21 cells along the temporal dimension.

Our results show that both models prefer the one-object hypothesis $H_1$ when presented with scenes L1 and L2 (Figure 1b). Since there are many more two-object scenes than one-object scenes, any typical two-object interpretation is assigned lower prior probability than a typical one-object interpretation. This preference for simpler interpretations is a consequence of the Bayesian Occam's razor. The baseline model makes the same kind of inference about scene U, and again prefers the one-object interpretation. Like infants, however, the smoothness model prefers the two-object interpretation of scene U. This model assigns low probability to a one-object interpretation where adjacent points on the object have very different velocities, and this preference for smooth motion is strong enough to overcome the simplicity preference that makes the difference when interpreting the other two scenes.

Other experiments from the developmental literature have produced results consistent with the principle of spatial smoothness. For example, 3.5-month olds are surprised when a tall object is fully hidden behind a short screen, 4 month olds are surprised when a large object appears to pass through a small slot, and 4.5-month olds expect a swinging screen to be interrupted when an object is placed in its path [1, 2]. All three inferences appear to rely on the expectation that objects tend not to shrink or to compress like foam rubber. Many of these experiments are consistent with an account that simply rules out non-rigid motion instead of introducing a graded preference for spatial smoothness. Biological motions, however, are typically non-rigid, and experiments suggest that infants can track and make inferences about objects that follow non-rigid trajectories [14]. Findings like these call for a theory like ours that incorporates a preference for rigid motion, but recognizes that non-rigid motions are possible.

## 3  Empirical findings: temporal smoothness

We now turn to the principle of temporal smoothness (S2) and discuss some of the experimental evidence that bears on this principle. Some researchers suggest that a closely related principle (inertia) is slow to develop, but we argue that expectations about temporal smoothness are needed to capture inferences made before the age of 4 months.

Baillargeon and DeVos [15] describe one relevant experiment that explores inferences about moving objects and obstacles. During habituation, 3.5-month-old infants saw a car pass behind an occluder and emerge from the other side (habituation stimulus H in Figure 2a). An obstacle was then placed in the direct path of the car (unlikely scenes U1 and U2) or beside this direct path (likely scene L), and the infants again saw the car pass behind the occluder and emerge from the other side. Looking

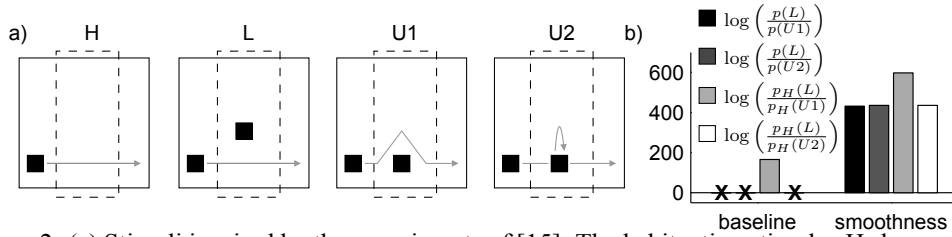

Figure 2: (a) Stimuli inspired by the experiments of [15]. The habituation stimulus H shows a block passing behind a barrier and emerging on the other side. After habituation, a new block is added either out of the direct path of the first block (L) or directly in the path of the first block (U1 and U2). In U1, the first block leaps over the second block, and in U2 the second block hops so that the first block can pass underneath. (b) Relative probabilities of scenes L, U1 and U2 according to two models. The baseline model finds all three scenes equally likely a priori, and considers L and U2 equally likely after habituation. The smoothness model considers L more likely than the other scenes both before and after habituation.

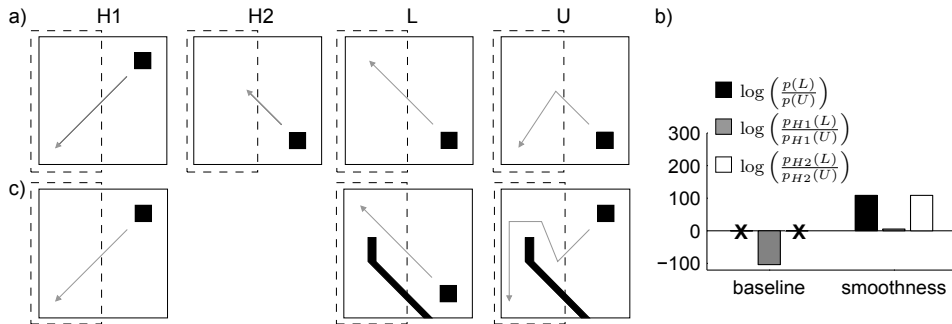

Figure 3: (a) Stimuli inspired by the experiments of Spelke et al. [16]. (b) Model predictions. After habituation to $H1$, the smoothness model assigns roughly equal probabilities to L and U. After habituation to $H2$, the model considers L more likely. (c) A stronger test of the inertia principle. Now the best interpretation of stimulus U involves multiple changes of direction.

time measurements suggested that the infants were more surprised to see the car emerge when the obstacle lay within the direct path of the car. This result is consistent with the principle of temporal smoothness, which suggests that infants expected the car to maintain a straight-line trajectory, and the obstacle to remain stationary.

We compared the smoothness model and the baseline model on a schematic version of this task. To model this experiment, we again assume that the infant's perceptual system has recovered a veridical velocity field, but now we must allow for occlusion. An ideal observer approach that treats a two dimensional scene as a projection of a three dimensional world can represent the occluder as an object in its own right. Here, however, we continue to work with a two dimensional world, and treat the occluded parts of the scene as missing data. An ideal observer approach should integrate over all possible values of the missing data, but for computational simplicity we approximate this approach by considering only one or two high-probability interpretations of each occluded scene.

We also need to account for habituation, and for cases where the habituation stimulus includes occlusion. We assume that an ideal observer computes a habituation field $\vec{v_H}$, or the velocity field with maximum posterior probability given the habituation stimulus. In Figure 2a, the inferred habituation field $\vec{v_H}$ specifies a trajectory where the block moves smoothly from the left to the right of the scene. We now assume that the observer expects subsequent velocity fields to be similar to $\vec{v_H}$. Formally, we use a product-of-experts approach to define a post-habituation distribution on velocity fields:

$$p_H(\vec{v}) \propto p(\vec{v})p(\vec{v}|\vec{v_H}) \tag{8}$$

The first expert $p(\vec{v})$ uses the prior distribution in Equation 6, and the second expert $p(\vec{v}|\vec{v_H})$ assumes that field $\vec{v}$ is drawn from a Gaussian distribution centered on $\vec{v_H}$. Intuitively, after habituation to $\vec{v_H}$ the second expert expects that subsequent velocity fields will be similar to $\vec{v_H}$. More information about this model of habituation is provided in the supporting material.

Given these assumptions, the black and dark gray bars in Figure 2 indicate relative a priori probabilities for scenes L, U1 and U2. The baseline model considers all three scenes equally probable,

but the smoothness model prefers L. After habituation, the baseline model is still unable to account for the behavioral data, since it considers scenes L and U2 to be equally probable. The smoothness model, however, continues to prefer L.

We previously mentioned three consequences of the principle of temporal smoothness: stationary objects tend to remain stationary, moving objects tend to keep moving, and moving objects tend to maintain a steady trajectory. The "car and obstacle" task addresses the first and third of these proposals, but other tasks provide support for the second. Many authors have studied settings where one moving object comes to a stop, and a second object starts to move [17]. Compared to the case where the first object collides with the second, infants appear to be surprised by the "no-contact" case where the two objects never touch. This finding is consistent with the temporal smoothness principle, which predicts that infants expect the first object to continue moving until forced to stop, and expect the second object to remain stationary until forced to start.

Other experiments [18] provide support for the principle of temporal smoothness, but there are also studies that appear inconsistent with this principle. In one of these studies [16], infants are initially habituated to a block that moves from one corner of an enclosure to another (H1 in Figure 3a). After habituation, infants see a block that begins from a different corner, and now the occluder is removed to reveal the block in a location consistent with a straight-line trajectory (L) or in a location that matches the final resting place during the habituation phase (U). Looking times suggest that infants aged 4-12 months are no more surprised by the inertia-violating outcome (U) than the inertia-consistent outcome (L). The smoothness model, however, can account for this finding. The outcome in U is contrary to temporal smoothness but consistent with habituation, and the tradeoff between these factors leads the model to assign roughly the same probability to scenes L and U (Figure 3b).

Only one of the inertia experiments described by Spelke et al. [16] and Spelke et al. [1] avoids this tradeoff between habituation and smoothness. This experiment considers a case where the habituation stimulus (H2 in Figure 3a) is equally similar to the two test stimuli. The results suggest that 8 month olds are now surprised by the inertia-violating outcome, and the predictions of our model are consistent with this finding (Figure 3b). 4 and 6 month olds, however, continue to look equally at the two outcomes. Note, however, that the trajectories in Figure 3 include at most one inflection point. Experiments that consider trajectories with many inflection points can provide a more powerful way of exploring whether 4 month olds have expectations about temporal smoothness.

One possible experiment is sketched in Figure 3c. The task is very similar to the task in Figure 3a, except that a barrier is added after habituation. In order for the block to end up in the same location as before, it must now follow a tortuous path around the barrier (U). Based on the principle of temporal smoothness, we predict that 4-month-olds will be more surprised to see the outcome in stimulus U than the outcome in stimulus L. This experimental design is appealing in part because previous work shows that infants are surprised by a case similar to U where the barrier extends all the way from one wall to the other [16], and our proposed experiment is a minor variant of this task.

Although there is room for debate about the status of temporal smoothness, we presented two reasons to revisit the conclusion that this principle develops relatively late. First, some version of this principle seems necessary to account for experiments like the car and obstacle experiment in Figure 2. Second, most of the inertia experiments that produced null results use a habituation stimulus which may have prevented infants from revealing their default expectations, and the one experiment that escapes this objection considers a relatively minor violation of temporal smoothness. Additional experiments are needed to explore this principle, but we predict that the inertia principle will turn out to be yet another example of knowledge that is available earlier than researchers once thought.

## 4   Discussion and Conclusion

We argued that characterizations of infant knowledge should include room for probabilistic expectations, and that probabilistic expectations about spatial and temporal smoothness appear to play a role in infant object perception. To support these claims we described an ideal observer model that includes both categorical (C1 through C5) and probabilistic principles (S1 and S2), and demonstrated that the categorical principles alone are insufficient to account for several experimental findings. Our two probabilistic principles are related to principles (rigidity and inertia) that have previously been described as categorical principles. Although rigidity and inertia appear to play a role in some early

inferences, formulating these principles as probabilistic expectations helps to explain how infants deal with non-rigid motion and violations of inertia.

Our analysis focused on some of the many existing experiments in the developmental literature, but new experiments will be needed to explore our probabilistic approach in depth. Categorical versions of a given principle (e.g. rigidity) allow room for only two kinds of behavior depending on whether the principle is violated or not. Probabilistic principles can be violated to a greater or lesser extent, and our approach predicts that violations of different magnitude may lead to different behaviors. Future studies of rigidity and inertia can consider violations of these principles that range from mild (Figure 3a) to severe (Figure 3c), and can explore whether infants respond to these violations differently. Future work should also consider whether the categorical principles we described (C1 through C5) are better characterized as probabilistic expectations. In particular, future studies can explore whether young infants consider large violations of cohesion (C5) or spatial continuity (C2) more surprising than smaller violations of these principles.

Although we did not focus on learning, our approach allows us to begin thinking formally about how principles of object perception might be acquired. First, we can explore how parameters like the smoothness parameters in our model ($\lambda_{\text{space}}$ and $\lambda_{\text{time}}$) might be tuned by experience. Second, we can use statistical model selection to explore transitions between different sets of principles. For instance, if a learner begins with the baseline model we considered (principles C1–C6), we can explore which subsequent observations provide the strongest statistical evidence for smoothness principles S1 and S2, and how much of this evidence is required before an ideal learner would prefer our smoothness model over the baseline model. It is not yet clear which principles of object perception could be learned, but the ideal observer approach can help to resolve this question.

## References

[1] E. S. Spelke, K. Breinlinger, J. Macomber, and K. Jacobson. Origins of knowledge. *Psychological Review*, 99:605–632, 1992.

[2] R. Baillargeon, L. Kotovsky, and A. Needham. The acquisition of physical knowledge in infancy. In D. Sperber, D. Premack, and A. J. Premack, editors, *Causal Cognition: A multidisciplinary debate*, pages 79–116. Clarendon Press, Oxford, 1995.

[3] E. S. Spelke. Principles of object perception. *Cognitive Science*, 14:29–56, 1990.

[4] E. Spelke. Initial knowledge: six suggestions. *Cognition*, 50:431–445, 1994.

[5] D. Mareschal and S. P. Johnson. Learning to perceive object unity: a connectionist account. *Developmental Science*, 5:151–172, 2002.

[6] D. Kersten and A. Yuille. Bayesian models of object perception. *Current opinion in Neurobiology*, 13: 150–158, 2003.

[7] Y. Weiss and E. H. Adelson. Slow and smooth: a Bayesian theory for the combination of local motion signals in human vision. Technical Report A.I Memo No. 1624, MIT, 1998.

[8] A. L. Yuille and N. M. Grzywacz. A mathematical analysis of the motion coherence theory. *International Journal of Computer Vision*, 3:155–175, 1989.

[9] W. S. Geisler. Physical limits of acuity and hyperacuity. *Journal of the Optical Society of America*, 1(7): 775–782, 1984.

[10] M. S. Banks and E. Shannon. Spatial and chromatic visual efficiency in human neonates. In *Visual perception and cognition in infancy*, pages 1–46. Lawrence Erlbaum Associates, Hillsdale, NJ, 1993.

[11] R. Baillargeon. Innate ideas revisited: for a principle of persistence in infants' physical reasoning. *Perspectives on Psychological Science*, 3(3):2–13, 2008.

[12] R. Kestenbaum, N. Termine, and E. S. Spelke. Perception of objects and object boundaries by three-month-old infants. *British Journal of Developmental Psychology*, 5:367–383, 1987.

[13] E. S. Spelke, C. von Hofsten, and R. Kestenbaum. Object perception and object-directed reaching in infancy: interaction of spatial and kinetic information for object boundaries. *Developmental Psychology*, 25:185–196, 1989.

[14] G. Huntley-Fenner, S. Carey, and A. Solimando. Objects are individuals but stuff doesn't count: perceived rigidity and cohesiveness influence infants' representations of small groups of discrete entities. *Cognition*, 85:203–221, 2002.

[15] R. Baillargeon and J. DeVos. Object permanence in young infants: further evidence. *Child Development*, 61(6):1227–1246, 1991.

[16] E. S. Spelke, G. Katz, S. E. Purcell, S. M. Ehrlich, and K. Breinlinger. Early knowledge of object motion: continuity and inertia. *Cognition*, 51:131–176, 1994.

[17] L. Kotovsky and R. Baillargeon. Reasoning about collisions involving inert objects in 7.5-month-old infants. *Developmental Science*, 3(3):344–359, 2000.

[18] T. Wilcox and A. Schweinle. Infants' use of speed information to individuate objects in occlusion events. *Infant Behavior and Development*, 26:253–282, 2003.
